# Information-Geometrical Significance of Sparsity in Gallager Codes

**Toshiyuki Tanaka**
Department of Electronics and Information Engineering
Tokyo Metropolitan University
Tokyo 192-0397, Japan
*tanaka@eei.metro-u.ac.jp*

**Shiro Ikeda**
Kyushu Institute of Technology & JST
Fukuoka 808-0196, Japan
*shiro@brain.kyutech.ac.jp*

**Shun-ichi Amari**
RIKEN, Brain Science Institute
Saitama 351-0198, Japan
*amari@brain.riken.go.jp*

## Abstract

We report a result of perturbation analysis on decoding error of the belief propagation decoder for Gallager codes. The analysis is based on information geometry, and it shows that the principal term of decoding error at equilibrium comes from the $m$-embedding curvature of the log-linear submanifold spanned by the estimated pseudoposteriors, one for the full marginal, and $K$ for partial posteriors, each of which takes a single check into account, where $K$ is the number of checks in the Gallager code. It is then shown that the principal error term vanishes when the parity-check matrix of the code is so sparse that there are no two columns with overlap greater than 1.

## 1  Introduction

Recent progress on error-correcting codes has attracted much attention because their decoders, exhibiting performance very close to Shannon's limit, can be implemented as neural networks. Important examples are turbo codes and Gallager codes [1]. It is now well understood that application of belief propagation to the respective graphical representations of the decoding problems for both codes yields practically efficient decoding algorithms which are the same as the existing ones (the turbo decoding [2] and the sum-product decoding [3], respectively). They are, however, not exact but approximate, since the associated graphical representations have loops in both cases. An important problem posed is to quantify the effect that comes from the existence of loops in the underlying graph. The so-called TAP approach [4] in statistical physics is an alternative way to formulate the same decoding algorithm [5]. Since this approach also assumes that the underlying graph is locally loop-free, one is faced with the same problem as above.

In this paper, we analyze the properties of the belief propagation decoder to Gallager codes, expecting that better theoretical understanding of the properties of the belief propagation

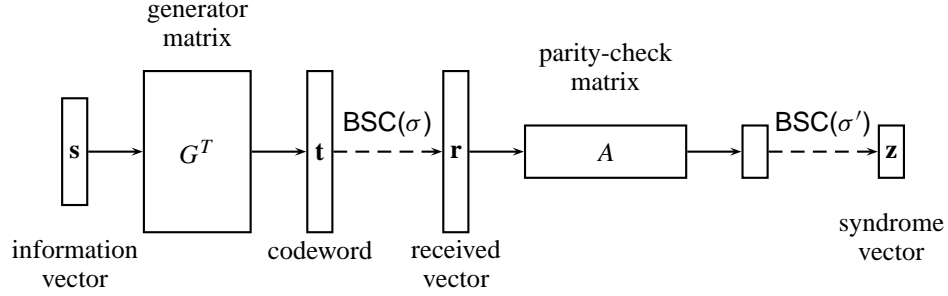

Figure 1: Gallager code

decoder will help understand the properties and efficiency of belief propagation in general, applied to loopy graphs, as well as those of the TAP approach. We specifically make use of the information geometry [6] and report a result of perturbation analysis on decoding error of the belief propagation decoder.

## 2 Gallager codes

Gallager code is defined by its parity-check matrix $A$, which has the form

$$A = [C_1 \mid C_2], \tag{1}$$

where $C_1$ and $C_2$ are $K \times M$ and $K \times K$ matrices, both of which are taken to be very sparse. $C_2$ is assumed invertible. We define the generator matrix of the Gallager code to be

$$G^T = \left[ \begin{array}{c} I \\ C_2^{-1} C_1 \end{array} \right] \tag{2}$$

where $I$ is the $M \times M$ identity matrix. $AG^T = O \bmod 2$ holds by definition.

The whole model of communication with the Gallager code is shown in Fig. 1. An information vector $\mathbf{s}$ of length $M$ is encoded into a codeword $\mathbf{t} = G^T \mathbf{s} \bmod 2$ of length $N \equiv K + M$. The codeword $\mathbf{t}$ is then transmitted over a channel. We assume that the transmission channel is a binary symmetric channel (BSC) with bit-error probability $\sigma$. The received vector is then $\mathbf{r} = \mathbf{t} + \mathbf{n} \bmod 2$, where $\mathbf{n}$ is the noise vector. Decoder tries to find the most probable $\mathbf{x}$ satisfying the parity-check equation

$$A\mathbf{x} = \mathbf{z} \bmod 2, \tag{3}$$

where $\mathbf{z} \equiv A\mathbf{r} \bmod 2$ is the syndrome vector. Since $A\mathbf{t} = AG^T \mathbf{s} = \mathbf{0} \bmod 2$, we have $\mathbf{z} = A\mathbf{n} \bmod 2$. Therefore, the solution $\mathbf{x}$ serves as an estimate of the noise vector $\mathbf{n}$. If we are successful in finding the true noise vector $\mathbf{n}$, we can reconstruct, from $\mathbf{r}$, the original codeword $\mathbf{t}$ by $\mathbf{t} = \mathbf{r} + \mathbf{n} \bmod 2$, and then the information vector $\mathbf{s}$. Since Eq. (3) is underdetermined, one has to take into account the prior knowledge of the noise in order to solve it properly.

The decoding problem can be cast into the Bayes framework. In the sequel, we transform expression of a bit from binary $(1, 0)$ to bipolar $(-1, 1)$. The prior for $\mathbf{x}$ is

$$p(\mathbf{x}) = \exp\big(\beta \mathbf{1} \cdot \mathbf{x} - N\psi(\beta)\big), \quad \psi(\beta) \equiv \log(e^{\beta} + e^{-\beta}), \tag{4}$$

where $\mathbf{1}$ is an $N$-dimensional vector whose elements are all 1, i.e., $\mathbf{1} \equiv [1, \ldots, 1]$. $\beta$ is a parameter which is related with the bit-error probability $\sigma$ of the transmission channel by

$$\sigma = \frac{1}{2}(1 - \tanh \beta). \tag{5}$$

For the sake of analytical tractability, we assume that the syndrome vector $\mathbf{z}$ is observed via another BSC channel with bit-error probability $\sigma'$ (see Fig. 1). This leads

$$p(\mathbf{z}|\mathbf{x}) \propto \exp\left[\rho \sum_{r=1}^{K} z_r \prod_{i \in \mathcal{L}(r)} x_i\right],$$
(6)

where $\mathcal{L}(r)$ is the set of all indices of nonzero elements in row $r$ of the parity-check matrix $A$, i.e., $\mathcal{L}(r) \equiv \{i \mid A_{ri} = 1\}$, and $\rho$ is defined by $\sigma' = (1/2)(1 - \tanh\rho)$. If we take the limit $\rho \to +\infty$, then we recover the conventional situation of observing the syndrome in a deterministic way. In what follows, we consider the case in which $\rho$ is finite, or equivalently, the case with soft parity-check constraints. Since experimental findings suggest that it is usually the case for decoding results of Gallager codes to violate no parity-check constraints [3], we believe that making the parity-check constraints soft does not alter essential properties of the problem.

## 3 Decoding

The posterior distribution of $\mathbf{x}$ for given observed syndrome $\mathbf{z}$ is derived from the prior $p(\mathbf{x})$ and the conditional $p(\mathbf{z} \mid \mathbf{x})$ by applying the Bayes formula, and the result is

$$p(\mathbf{x}|\mathbf{z}) \propto \exp\left[c_0(\mathbf{x}) + \rho \sum_{r=1}^{K} c_r(\mathbf{x})\right],$$
(7)

where we let

$$c_0(\mathbf{x}) \equiv \beta \mathbf{1} \cdot \mathbf{x}, \quad c_r(\mathbf{x}) \equiv z_r \prod_{i \in \mathcal{L}(r)} x_i \quad (r = 1, \ldots, K).$$
(8)

The objective of decoder of Gallager codes is to obtain the marginal-posterior-mode (MPM) estimate based on the posterior $p(\mathbf{x}|\mathbf{z})$:

$$\hat{x}_i = \arg\max_{x_i} \sum_{\mathbf{x} \backslash x_i} p(\mathbf{x}|\mathbf{z}).$$
(9)

The MPM estimation provides the Bayes-optimum decoder minimizing expected bit-error probability of the decoding results. However, the marginalization is in general computationally hard, which renders the decoding problem intractable. An approximate decoding algorithm, originally proposed by Gallager himself [1], is known to be efficient in practice. It has been recently rediscovered by MacKay [3] by application of the belief propagation to the underlying graphical model. Murayama et al. [5] also formulated the same algorithm based on the so-called TAP approach [4]. The decoder implementing the algorithm is called the belief propagation decoder, and is analyzed in the following.

We define a family of distributions with a set of parameters $\boldsymbol{\zeta} = (\zeta_1, \ldots, \zeta_N)^T$ and $\mathbf{v} = (v_1, \ldots, v_K)$:

$$S = \left\{p(\mathbf{x}; \boldsymbol{\zeta}, \mathbf{v}) \mid p(\mathbf{x}; \boldsymbol{\zeta}, \mathbf{v}) = \exp\left[\boldsymbol{\zeta} \cdot \mathbf{x} + \mathbf{v} \cdot \mathbf{c}(\mathbf{x}) - \varphi(\boldsymbol{\zeta}, \mathbf{v})\right]\right\},$$
(10)

where $\mathbf{c}(\mathbf{x}) \equiv \left(c_1(\mathbf{x}), \ldots, c_K(\mathbf{x})\right)^T$. The family $S$ includes the factorizable test distribution $p_0(\mathbf{x}; \boldsymbol{\theta}_0)$ $(= p(\mathbf{x}; \boldsymbol{\theta}_0, \mathbf{0}))$, the true posterior $p(\mathbf{x}|\mathbf{z})$ $(= p(\mathbf{x}; \beta\mathbf{1}, \rho\mathbf{1}))$, and $K$ partial posteriors $p_r(\mathbf{x}; \boldsymbol{\theta}_r)$ $(= p(\mathbf{x}; \boldsymbol{\theta}_r, \rho\mathbf{e}_r))$; $\mathbf{e}_r \equiv (0, \ldots, 0, \underset{\hat{r}}{1}, 0, \ldots, 0)^T$).

We then define the expectation parameter $\boldsymbol{\eta}(\boldsymbol{\zeta}, \mathbf{v})$ by

$$\boldsymbol{\eta}(\boldsymbol{\zeta}, \mathbf{v}) \equiv \sum_{\mathbf{x}} \mathbf{x}\, p(\mathbf{x}; \boldsymbol{\zeta}, \mathbf{v}).$$
(11)

The marginalization in Eq. (9) corresponds to evaluating the expectation parameter of the true posterior. We now introduce the equimarginal family

$$M(\boldsymbol{\theta}_0) \equiv \big\{ p(\mathbf{x}; \boldsymbol{\zeta}, \mathbf{v}) \mid \boldsymbol{\eta}(\boldsymbol{\zeta}, \mathbf{v}) = \boldsymbol{\eta}(\boldsymbol{\theta}_0, \mathbf{0}) \big\}, \tag{12}$$

and define the marginalization operator $\Pi$ as follows: For $p \in S$, $\Pi \circ p \equiv \boldsymbol{\theta}_0$ if $p \in M(\boldsymbol{\theta}_0)$. Knowing $\boldsymbol{\theta}_0 = \Pi \circ p$ is regarded as being equivalent to knowing the expectation parameter of $p$, since $\boldsymbol{\eta}(\boldsymbol{\theta}_0, \mathbf{0})$ is easily evaluated from $\boldsymbol{\theta}_0$; in other words, the marginalization is tractable for distributions belonging to the factorizable model:

$$M_0 \equiv \big\{ p_0(\mathbf{x}; \boldsymbol{\theta}_0) \equiv p(\mathbf{x}; \boldsymbol{\theta}_0, \mathbf{0}) = \exp(\boldsymbol{\theta}_0 \cdot \mathbf{x} - \varphi_0(\boldsymbol{\theta}_0)) \big\} \tag{13}$$

The basic assumption of iterative decoding is that the marginalization is also tractable for the models corresponding to constituent decoders with single checks, with factorizable priors:

$$M_r \equiv \big\{ p_r(\mathbf{x}; \boldsymbol{\theta}) \equiv p(\mathbf{x}; \boldsymbol{\theta}, \rho \mathbf{e}_r) = \exp(\boldsymbol{\theta} \cdot \mathbf{x} + \rho c_r(\mathbf{x}) - \varphi_r(\boldsymbol{\theta})) \big\} \tag{14}$$

The algorithm of the belief propagation decoder is described in the notation employed here as follows:

**Initialization:**  Let $t = 0$ and $\boldsymbol{\theta}_r^0 = \beta \mathbf{1}$, $r = 1, \dots, K$.

**Horizontal step:**  Evaluate the marginalization of $p_r(\mathbf{x}; \boldsymbol{\theta}_r^t) \in M_r$ to produce a guess $\boldsymbol{\zeta}_r^t$ based on the current prior information $\boldsymbol{\theta}_r^t$ and the check $z_r$:

$$\boldsymbol{\zeta}_r^t = \Pi \circ p_r(\mathbf{x}; \boldsymbol{\theta}_r^t), \quad r = 1, \dots, K, \tag{15}$$

and calculate a net contribution (the 'cavity field' [7]) from the check $z_r$ by subtracting the prior information:

$$\boldsymbol{\xi}_r^t = \boldsymbol{\zeta}_r^t - \boldsymbol{\theta}_r^t. \tag{16}$$

(It should be noted that $(\boldsymbol{\xi}_r^t)_i = 0$ holds for $i \notin \mathcal{L}(r)$ as it should be, since the constituent decoder with check $z_r$ cannot provide any contribution regarding information of $x_i$, $i \notin \mathcal{L}(r)$.)

**Vertical step:**  Compose the 'leave-one-out' estimates [7]

$$\boldsymbol{\theta}_r^{t+1} = \beta \mathbf{1} + \sum_{r' \neq r} \boldsymbol{\xi}_{r'}^t, \quad r = 1, \dots, K, \tag{17}$$

and the pseudoposterior

$$\boldsymbol{\theta}^{t+1} = \beta \mathbf{1} + \sum_{r=1}^{K} \boldsymbol{\xi}_r^t. \tag{18}$$

Iterate the above steps until convergence is achieved. The desired decoding result $\boldsymbol{\eta}(\beta \mathbf{1}, \rho \mathbf{1})$ is then approximated by $\boldsymbol{\eta}(\boldsymbol{\theta}^*, \mathbf{0})$, where $\boldsymbol{\theta}^*$ is the convergent value of $\{\boldsymbol{\theta}^t\}$.

## 4   Information-geometrical characterization of equilibrium

Assume that the convergence is achieved and let $(\boldsymbol{\theta}^*, \boldsymbol{\xi}_1^*, \dots, \boldsymbol{\xi}_K^*)$ be the equilibrium values of $(\boldsymbol{\theta}^t, \boldsymbol{\xi}_1^t, \dots, \boldsymbol{\xi}_K^t)$. Then, from Eqs. (15) and (16), we have

$$\Pi \circ p_r(\mathbf{x}; \boldsymbol{\theta}^* - \boldsymbol{\xi}_r^*) = \boldsymbol{\theta}^*, \quad r = 1, \dots, K. \tag{19}$$

This means that $p_0(\mathbf{x}; \boldsymbol{\theta}^*)$ and $p_r(\mathbf{x}; \boldsymbol{\theta}^* - \boldsymbol{\xi}_r^*)$, $r = 1, \dots, K$, are equimarginal, that is,

$$p_r(\mathbf{x}; \boldsymbol{\theta}^* - \boldsymbol{\xi}_r^*) \in M(\boldsymbol{\theta}^*), \quad r = 1, \dots, K \tag{20}$$

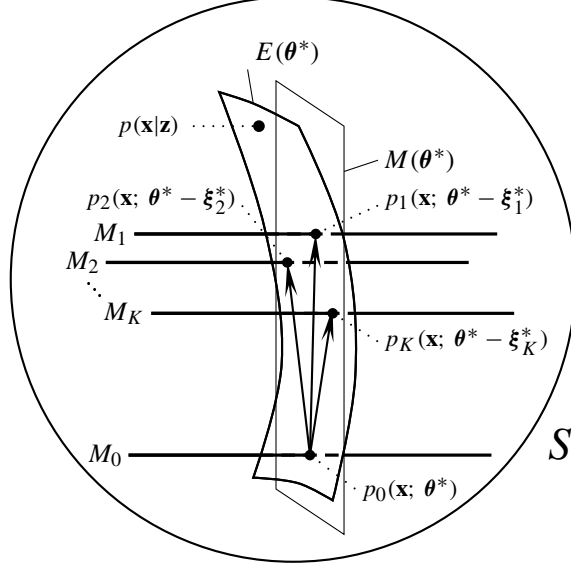

Figure 2: Geometric structure of belief propagation decoder

holds. Another property of the equilibrium is the log-linear relation

$$\log p(\mathbf{x}|\mathbf{z}) - \log p_0(\mathbf{x};\,\boldsymbol{\theta}^*) = \sum_{r=1}^{K}\left\{\log p_r(\mathbf{x};\,\boldsymbol{\theta}^* - \boldsymbol{\xi}_r^*) - \log p_0(\mathbf{x};\,\boldsymbol{\theta}^*)\right\} + \text{const.} \quad (21)$$

or, in the $(\boldsymbol{\zeta},\,\mathbf{v})$ coordinate,

$$(\beta\mathbf{1},\,\rho\mathbf{1}) - (\boldsymbol{\theta}^*,\,\mathbf{0}) = \sum_{r=1}^{K}\left((\boldsymbol{\theta}^* - \boldsymbol{\xi}_r^*,\,\rho\mathbf{e}_r) - (\boldsymbol{\theta}^*,\,\mathbf{0})\right). \quad (22)$$

This means that the true posterior $p(\mathbf{x}|\mathbf{z})$ belongs to the 'log-linear submanifold' $E(\boldsymbol{\theta}^*)$, the affine subspace in the $(\boldsymbol{\zeta},\,\mathbf{v})$-coordinate rooted at $(\boldsymbol{\theta}^*,\,\mathbf{0})$ and spanned by $(-\boldsymbol{\xi}_r^*,\,\rho\mathbf{e}_r)$, $r = 1,\ldots,K$.

These two properties do *not* imply $p(\mathbf{x}|\mathbf{z}) \in M(\boldsymbol{\theta}^*)$. In fact, if we *were* to assume, instead of the log-linear relation (21), the linear relation

$$p(\mathbf{x}|\mathbf{z}) - p_0(\mathbf{x};\,\boldsymbol{\theta}^*) = \sum_{r=1}^{K}\left\{p_r(\mathbf{x};\,\boldsymbol{\xi}_r^*) - p_0(\mathbf{x};\,\boldsymbol{\theta}^*)\right\}, \quad (23)$$

then we would have $p(\mathbf{x}|\mathbf{z}) \in M(\boldsymbol{\theta}^*)$ and thus $\boldsymbol{\eta}(\beta\mathbf{1},\,\rho\mathbf{1}) = \boldsymbol{\eta}(\boldsymbol{\theta}^*,\,\mathbf{0})$. This is not the case because of the difference between the linear and log-linear relations. To what degree the log-linear relation deviates from the linear relation determines the decoding error of belief propagation decoder. The structure is best described on the basis of information geometry [6]. Figure 2 illustrates the geometric structure of the belief propagation decoder. It should be noted that the geometrical structure shown here is essentially the same as that for the turbo decoding [8, 9].

## 5 Main result

Based on the information geometry, we have evaluated decoding error, the difference between the true expectation $\boldsymbol{\eta}(\beta\mathbf{1},\,\rho\mathbf{1})$ and its estimate by the belief propagation decoder

$\eta(\theta^*, \mathbf{0})$, via perturbation analysis. Taking into account the terms up to second order, we have

$$\eta(\beta\mathbf{1}, \rho\mathbf{1}) - \eta(\theta^*, \mathbf{0}) = \frac{\rho^2}{2} \sum_{r,s;r\neq s} B_{rs}\eta(\theta^*, \mathbf{0}) + O(\rho^3), \tag{24}$$

where

$$B_{rs} \equiv \left(\frac{\partial}{\partial v_r} - \sum_{k=1}^{N} g^{kk} A_r^k \frac{\partial}{\partial \theta_k}\right)\left(\frac{\partial}{\partial v_s} - \sum_{j=1}^{N} g^{jj} A_s^j \frac{\partial}{\partial \theta_j}\right), \tag{25}$$

and

$$A_r^i \equiv \frac{\partial \eta_i(\theta^*, \mathbf{0})}{\partial v_r} = \mathrm{Cov}_{\theta^*,\mathbf{0}}\big[x_i, \ c_r(\mathbf{x})\big]. \tag{26}$$

$\{B_{rs}\}$ are the elements of the $m$-embedding curvature tensor of the manifold $E(\theta^*)$ in $S$. $g^{ii} \equiv 1/(1 - \eta_{ii}(\theta^*, \mathbf{0})^2)$ are the diagonal elements of the inverse of the Fisher information of $p_0(\mathbf{x}; \theta^*)$. This is the generalization of the result obtained for the turbo decoding [8].

Explicit calculation gives the following theorem.

**Theorem 1.** The decoding error of belief propagation decoder is given, within the second-order with respect to $\rho$, by

$$\eta_i(\beta\mathbf{1}, \rho\mathbf{1}) - \eta_i(\theta^*, \mathbf{0})$$

$$= \rho^2(1 - \eta_i^2)\Bigg[-\eta_i \sum_{\substack{r,s \\ r\neq s, i\in\mathcal{L}(r)\cap\mathcal{L}(s)}} z_r z_s \sum_{j\in(\mathcal{L}(r)\cap\mathcal{L}(s))\backslash i} (1 - \eta_j^2)$$

$$\times \prod_{k\in(\mathcal{L}(r)\cap\mathcal{L}(s))\backslash i,j} \eta_k^2 \prod_{l\in(\mathcal{L}(r)-\mathcal{L}(s))\cup(\mathcal{L}(s)-\mathcal{L}(r))} \eta_l$$

$$+ \sum_{\substack{r,s \\ r\neq s, i\in\mathcal{L}(r)-\mathcal{L}(s)}} z_r z_s \Bigg(1 - \prod_{j\in\mathcal{L}(r)\cap\mathcal{L}(s)} \eta_j^2 - \sum_{j\in\mathcal{L}(r)\cap\mathcal{L}(s)} (1 - \eta_j^2) \prod_{k\in(\mathcal{L}(r)\cap\mathcal{L}(s))\backslash j} \eta_k^2\Bigg)$$

$$\times \prod_{l\in[(\mathcal{L}(r)-\mathcal{L}(s))\backslash i]\cup[\mathcal{L}(s)-\mathcal{L}(r)]} \eta_l\Bigg]$$

$$+ O(\rho^3) \tag{27}$$

where $\eta_i \equiv \eta_i(\theta^*, \mathbf{0})$.

From this theorem, it immediately follows that:

**Corollary 2.** If the parity-check matrix $A$ has no two columns with overlap greater than 1, then the principal error term, given in Eq. (27), vanishes.

These are the main result of this paper.

## 6 Discussion

The general result given in Eq. (24) shows that the principal error term is not coordinate invariant, since the summation with respect to $r$ and $s$ in the right-hand side of Eq. (24) excludes terms with $r = s$. This corresponds to the empirical fact that the performance does depend on the design of the code, that is, the choice of the parity-check matrix $A$. Explicit evaluation of the principal error term, as in Theorem 1, makes it possible to improve the

performance of a code, just in the same way as the perturbational approach to improving the naive mean-field approximation [10, 11, 12, 13, 14, 15, 16, 17].

It is believed [3] that Gallager codes have smaller average probability of decoding error if we avoid any two columns of the parity-check matrix $A$ to have overlap greater than 1. An intuitive explanation to this belief is that such avoidance prevents loops with length 4 from appearing in the graphical representation. Since short loops are expected to do harm in proper functioning of belief propagation, their existence may raise the possibility of decoding errors. Our result supports this belief by showing analytically that the principal term of decoding error vanishes when the parity-check matrix of the code is so sparse and prepared with care so that there are no two columns with overlap greater than 1.

Loops with length longer than 4 do not contribute to the decoding error at least via the principal term, but they may have effects via higher-order terms. Our analysis presented here can be extended in a straightforward manner to higher-order perturbation analysis in order to quantify these effects.

It should be noted that our approach taken in this paper is different from the common approach to analyzing the properties of the belief propagation decoder in the literature, in that we do not consider *ensembles* of codes. A typical reasoning found in the literature (e.g., [18]) is first to consider an ensemble of random parity-check matrices, to state that the probability (over the ensemble) of containing short loops in the associated graph decreases down to zero as the size of the parity-check matrix tends to infinity, and to assume that the behavior of the belief propagation decoder for codes with longer loops is the same as that of belief propagation for loop-free case. The statistical-mechanical approach to performance analysis of Gallager-type codes [5] also assumes random ensembles. Our analysis, on the other hand, does not assume ensembles but allows, although asymptotically, performance evaluation of the belief propagation decoder to Gallager codes with any *single instance* of the parity-check matrix with finite size.

## Acknowledgments

The authors would like to thank Dr. Yoshiyuki Kabashima for his helpful suggestions and comments.

## References

[1]  R. G. Gallager, *Low Density Parity Check Codes*, Ph. D. Thesis, Mass. Inst. Tech., 1960.

[2]  R. J. McEliece, D. J. C. MacKay, and J. Cheng, "Turbo decoding as an instance of Pearl's `belief propagation' algorithm," *IEEE J. Select. A. Commun.*, vol. 16, no. 2, pp. 140–152, 1998.

[3]  D. J. C. MacKay, "Good error-correcting codes based on very sparse matrices," *IEEE Trans. Inform. Theory,* vol. 45, no. 2, pp. 399–431, 1999.

[4]  D. J. Thouless, P. W. Anderson, and R. G. Palmer, "Solution of `Solvable model of a spin glass'," *Phil. Mag.*, vol. 35, no. 3, pp. 593–601, 1977.

[5]  T. Murayama, Y. Kabashima, D. Saad, and R. Vicente, "Statistical physics of regular low-density parity-check error-correcting codes," *Phys. Rev. E*, vol. 62, no. 2, pp. 1577–1591, 2000.

[6]  S. Amari and H. Nagaoka (Transl. by D. Harada), *Methods of Information Geometry*, Translations of Mathematical Monographs, vol. 191, American Math. Soc., 2000.

[7]  Y. Kabashima and D. Saad, "The TAP approach to intensive and extensive connectivity systems," in M. Opper and D. Saad (eds.), *Advanced Mean Field Methods — Theory and Practice*, The MIT Press, 2001, pp. 65–84.

[8]  S. Ikeda, T. Tanaka, and S. Amari, "Information geometrical framework for analyzing belief propagation decoder," in T. G. Dietterich *et al.* (eds.), *Advances in Neural Information Processing Systems*, vol. 14 (this volume), The MIT Press, 2002.

[9] S. Ikeda, T. Tanaka, and S. Amari, "Information geometry of turbo codes and low-density parity-check codes," submitted to *IEEE Trans. Inform. Theory*, 2001.

[10] H. J. Kappen and F. B. Rodriguez, "Efficient learning in Boltzmann machines using linear response theory," *Neural Computation*, vol. 10, no. 5, pp. 1137–1156, 1998.

[11] H. J. Kappen and F. B. Rodriguez, "Boltzmann machine learning using mean field theory and linear response correction," in M. I. Jordan *et al.* (eds.), *Advances in Neural Information Processing Systems*, vol. 10, The MIT Press, 1998, pp. 280–286.

[12] T. Tanaka, "A theory of mean field approximation," in M. S. Kearns *et al.* (eds.), *Advances in Neural Information Processing Systems*, vol. 11, The MIT Press, 1999, pp. 351–357.

[13] T. Tanaka, "Information geometry of mean-field approximation," *Neural Computation*, vol. 12, no. 8, pp. 1951–1968, 2000.

[14] J. S. Yedidia, "An idiosyncratic journey beyond mean field theory," in M. Opper and D. Saad (eds.), *Advanced Mean Field Methods — Theory and Practice*, The MIT Press, 2001, pp. 21–35.

[15] H. J. Kappen and W. J. Wiegerinck, "Mean field theory for graphical models," in M. Opper and D. Saad (eds.), *Advanced Mean Field Methods — Theory and Practice*, The MIT Press, 2001, pp. 37–49.

[16] S. Amari, S. Ikeda, and H. Shimokawa, "Information geometry of $\alpha$-projection in mean field approximation," in M. Opper and D. Saad (eds.), *Advanced Mean Field Methods — Theory and Practice*, The MIT Press, 2001, pp. 241–257.

[17] T. Tanaka, "Information geometry of mean-field approximation," in M. Opper and D. Saad (eds.), *Advanced Mean Field Methods — Theory and Practice*, The MIT Press, 2001, pp. 259–273.

[18] T. J. Richardson and R. L. Urbanke, "The capacity of low-density parity-check codes under message-passing decodeing," *IEEE Trans. Inform. Theory*, vol. 47, no. 2, pp. 599–618, 2001.
